# Dynamic Bayesian Networks with Deterministic Latent Tables

**David Barber**
Institute for Adaptive and Neural Computation
Edinburgh University
5 Forrest Hill, Edinburgh, EH1 2QL, U.K.
dbarber@anc.ed.ac.uk

## Abstract

The application of latent/hidden variable Dynamic Bayesian Networks is constrained by the complexity of marginalising over latent variables. For this reason either small latent dimensions or Gaussian latent conditional tables linearly dependent on past states are typically considered in order that inference is tractable. We suggest an alternative approach in which the latent variables are modelled using deterministic conditional probability tables. This specialisation has the advantage of tractable inference even for highly complex non-linear/non-Gaussian visible conditional probability tables. This approach enables the consideration of highly complex latent dynamics whilst retaining the benefits of a tractable probabilistic model.

## 1 Introduction

Dynamic Bayesian Networks are a powerful framework for temporal data models with widespread application in time series analysis[10, 2, 5]. A time series of length $T$ is a sequence of observation vectors $\mathcal{V} = \{\mathbf{v}(1), \mathbf{v}(2), \ldots, \mathbf{v}(T)\}$, where $v_i(t)$ represents the state of visible variable $i$ at time $t$. For example, in a speech application $\mathcal{V}$ may represent a vector of cepstral coefficients through time, the aim being to classify the sequence as belonging to a particular phonene[2, 9]. The power in the Dynamic Bayesian Network is the assumption that the observations may be generated by some latent (hidden) process that cannot be directly experimentally observed. The basic structure of these models is shown in fig(1)[a] where network states are only dependent on a short time history of previous states (the Markov assumption). Representing the hidden variable sequence by $\mathcal{H} = \{\mathbf{h}(1), \mathbf{h}(2), \ldots, \mathbf{h}(T)\}$, the joint distribution of a first order Dynamic Bayesian Network is

$$p(\mathcal{V}, \mathcal{H}) = p(\mathbf{v}(1))p(\mathbf{h}(1)|\mathbf{v}(1)) \prod_{t=1}^{T-1} p(\mathbf{v}(t+1)|\mathbf{v}(t), \mathbf{h}(t))p(\mathbf{h}(t+1)|\mathbf{v}(t), \mathbf{v}(t+1), \mathbf{h}(t))$$

This is a Hidden Markov Model (HMM), with additional connections from visible to hidden units[9]. The usage of such models is varied, but here we shall concentrate on unsupervised sequence learning. That is, given a set of training sequences

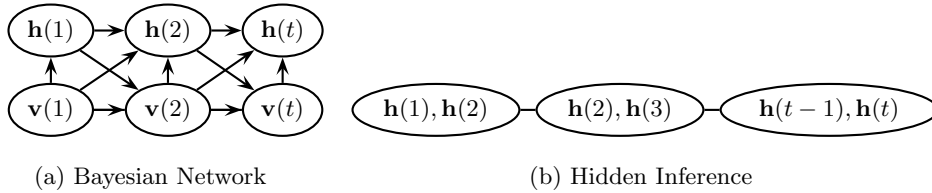

(a) Bayesian Network                        (b) Hidden Inference

Figure 1: (a) A first order Dynamic Bayesian Network containing a sequence of hidden (latent) variables $\mathbf{h}(1), \mathbf{h}(2), \ldots, \mathbf{h}(T)$ and a sequence of visible (observable) variables $\mathbf{v}(1), \mathbf{v}(2), \ldots, \mathbf{v}(T)$. In general, all conditional probability tables are stochastic – that is, more than one state can be realised. (b) Conditioning on the visible units forms an undirected chain in the hidden space. Hidden unit inference is achieved by propagating information along both directions of the chain to ensure normalisation.

$\mathcal{V}^1, \ldots, \mathcal{V}^P$ we aim to capture the essential features of the underlying dynamical process that generated the data. Denoting the parameters of the model by $\boldsymbol{\Theta}$, learning can be achieved using the EM algorithm which maximises a lower bound on the likelihood of a set of observed sequences by the procedure[5]:

$$\boldsymbol{\Theta}^{new} = \arg \max_{\boldsymbol{\Theta}} \sum_{\mu=1}^{P} p(\mathcal{H}^\mu | \mathcal{V}^\mu, \boldsymbol{\Theta}^{old}) \log p(\mathcal{H}^\mu, \mathcal{V}^\mu, \boldsymbol{\Theta}). \tag{1}$$

This procedure contains expectations with respect to the distribution $p(\mathcal{H}|\mathcal{V})$ – that is, to do learning, we need to infer the hidden unit distribution conditional on the visible variables. $p(\mathcal{H}|\mathcal{V})$ is represented by the undirected clique graph, fig(1)[b], in which each node represents a function (dependent on the clamped visible units) of the hidden variables it contains, with $p(\mathcal{H}|\mathcal{V})$ being the product of these clique potentials. In order to do inference on such a graph, in general, it is necessary to carry out a message passing type procedure in which messages are first passed one way along the undirected graph, and then back, such as in the forward-backward algorithm in HMMs [5]. Only when messages have been passed along both directions of all links can the normalised conditional hidden unit distribution be numerically determined. The complexity of calculating messages is dominated by marginalisation of the clique functions over a hidden vector $\mathbf{h}(t)$. In the case of discrete hidden units with $S$ states, this complexity is of the order $S^2$, and the total complexity of inference is then $O(TS^2)$. For continuous hidden units, the analogous marginalisation requires integration of a clique function over a hidden vector. If the clique function is very low dimensional, this may be feasible. However, in high dimensions, this is typically intractable unless the clique functions are of a very specific form, such as Gaussians. This motivates the Kalman filter model[5] in which all conditional probability tables are Gaussian with means determined by a linear combination of previous states. There have been several attempts to generalise the Kalman filter to include non-linear/non-Gaussian conditional probability tables, but most rely on using approximate integration methods based on either sampling[3], perturbation or variational type methods[5].

In this paper we take a different approach. We consider specially constrained networks which, when conditioned on the visible variables, render the hidden unit

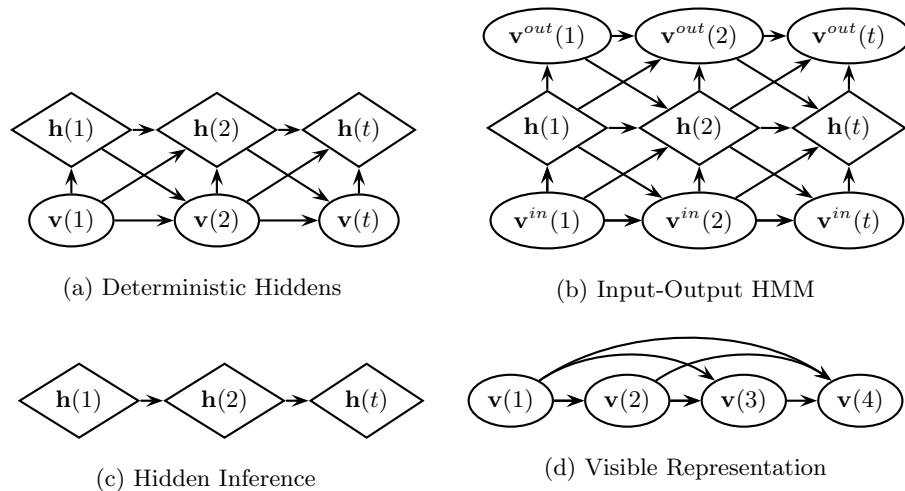

(a) Deterministic Hiddens    (b) Input-Output HMM

(c) Hidden Inference    (d) Visible Representation

Figure 2: (a) A first order Dynamic Bayesian Network with deterministic hidden CPTs (represented by diamonds) – that is, the hidden node is certainly in a single state, determined by its parents. (b) An input-output HMM with deterministic hidden variables. (c) Conditioning on the visible variables forms a directed chain in the hidden space which is deterministic. Hidden unit inference can be achieved by forward propagation alone. (d) Integrating out hidden variables gives a cascade style directed visible graph, shown here for only four time steps.

distribution trivial. The aim is then to be able to consider non-Gaussian and non-linear conditional probability tables (CPTs), and hence richer dynamics in the hidden space.

## 2   Deterministic Latent Variables

The deterministic latent CPT case, fig(2)[a] defines conditional probabilities

$$p(\mathbf{h}(t+1)|\mathbf{v}(t+1),\mathbf{v}(t),\mathbf{h}(t)) = \boldsymbol{\delta}\left(\mathbf{h}(t+1) - \mathbf{f}\left(\mathbf{v}(t+1),\mathbf{v}(t),\mathbf{h}(t),\boldsymbol{\theta}_{\mathbf{h}}\right)\right) \quad (2)$$

where $\boldsymbol{\delta}(x)$ represents the Dirac delta function for continuous hidden variables, and the Kronecker delta for discrete hidden variables. The vector function $\mathbf{f}$ parameterises the CPT, itself having parameters $\boldsymbol{\theta}_{\mathbf{h}}$. Whilst the restriction to deterministic CPTs appears severe, the model retains some attractive features : The marginal $p(\mathcal{V})$ is non-Markovian, coupling all the variables in the sequence, see fig(2)[d]. The marginal $p(\mathcal{H})$ is stochastic, whilst hidden unit inference is deterministic, as illustrated in fig(2)[c]. Although not considered explicitly here, input-output HMMs[7], see fig(2)[b], are easily dealt with by a trivial modification of this framework.

For learning, we can dispense with the EM algorithm and calculate the log likelihood of a single training sequence $\mathcal{V}$ directly,

$$L(\boldsymbol{\theta}_{\mathbf{v}},\boldsymbol{\theta}_{\mathbf{h}}|\mathcal{V}) = \log p(\mathbf{v}(1)|\boldsymbol{\theta}_{\mathbf{v}}) + \sum_{t=1}^{T-1} \log p(\mathbf{v}(t+1)|\mathbf{v}(t),\mathbf{h}(t),\boldsymbol{\theta}_{\mathbf{v}}) \quad (3)$$

where the hidden unit values are calculated recursively using

$$\mathbf{h}(t+1) = \mathbf{f}\left(\mathbf{v}(t+1), \mathbf{v}(t), \mathbf{h}(t), \boldsymbol{\theta}_\mathbf{h}\right) \tag{4}$$

The adjustable parameters of the hidden and visible CPTs are represented by $\boldsymbol{\theta}_\mathbf{h}$ and $\boldsymbol{\theta}_\mathbf{v}$ respectively. The case of training multiple independently generated sequences $\mathcal{V}^\mu, \mu = 1, \ldots P$ is straightforward and has likelihood $\sum_\mu L(\boldsymbol{\theta}_\mathbf{v}, \boldsymbol{\theta}_\mathbf{h} | \mathcal{V}^\mu)$. To maximise the log-likelihood, it is useful to evaluate the derivatives with respect to the model parameters. These can be calculated as follows :

$$\frac{dL}{d\boldsymbol{\theta}_\mathbf{v}} = \frac{\partial \log p(\mathbf{v}(1)|\boldsymbol{\theta}_\mathbf{v})}{\partial \boldsymbol{\theta}_\mathbf{v}} + \sum_{t=1}^{T-1} \frac{\partial}{\partial \boldsymbol{\theta}_\mathbf{v}} \log p(\mathbf{v}(t+1)|\mathbf{v}(t), \mathbf{h}(t), \boldsymbol{\theta}_\mathbf{v}) \tag{5}$$

$$\frac{dL}{d\boldsymbol{\theta}_\mathbf{h}} = \sum_{t=1}^{T-1} \frac{\partial}{\partial \mathbf{h}(t)} \log p(\mathbf{v}(t+1)|\mathbf{v}(t), \mathbf{h}(t), \boldsymbol{\theta}_\mathbf{v}) \frac{d\mathbf{h}(t)}{d\boldsymbol{\theta}_\mathbf{h}} \tag{6}$$

$$\frac{d\mathbf{h}(t)}{d\boldsymbol{\theta}_\mathbf{h}} = \frac{\partial \mathbf{f}(t)}{\partial \boldsymbol{\theta}_\mathbf{h}} + \frac{\partial \mathbf{f}(t)}{\partial \mathbf{h}(t-1)} \frac{d\mathbf{h}(t-1)}{d\boldsymbol{\theta}_\mathbf{h}} \tag{7}$$

where $\mathbf{f}(t) \equiv \mathbf{f}(\mathbf{v}(t), \mathbf{v}(t-1), \mathbf{h}(t-1), \boldsymbol{\theta}_\mathbf{h})$. Hence the derivatives can be calculated by deterministic forward propagation of errors and highly complex functions $\mathbf{f}$ and CPTs $p(\mathbf{v}(t+1)|\mathbf{v}(t), \mathbf{h}(t))$ may be used. Whilst the training of such networks resembles back-propagation in neural networks [1, 6], the models have a stochastic interpretation and retain the benefits inherited from probability theory, including the possibility of a Bayesian treatment.

## 3 A Discrete Visible Illustration

To make the above framework more explicit, we consider the case of continuous hidden units and discrete, binary visible units, $v_i(t) \in \{0, 1\}$. In particular, we restrict attention to the model:

$$p(\mathbf{v}(t+1)|\mathbf{v}(t), \mathbf{h}(t)) = \prod_{i=1}^{V} \sigma\left((2v_i(t+1) - 1)\sum_j w_{ij}\phi_j(t)\right), \quad h_i(t+1) = \sum_j u_{ij}\psi_j(t)$$

where $\sigma(x) = 1/(1 + e^{-x})$ and $\phi_j(t)$ and $\psi_j(t)$ represent fixed functions of the network state $(\mathbf{h}(t), \mathbf{v}(t))$. Normalisation is ensured since $1 - \sigma(x) = \sigma(-x)$. This model generalises a recurrent stochastic heteroassociative Hopfield network[4] to include deterministic hidden units dependent on past network states.

The derivatives of the log likelihood are given by :

$$\frac{dL}{dw_{ij}} = \sum_t (1 - \sigma_i(t))(2v_i(t+1) - 1)\phi_j(t), \quad \frac{dL}{du_{ij}} = \sum_{t,k,l}(1 - \sigma_k(t))(2v_k(t+1) - 1)w_{kl}\phi_l'(t)\frac{dh_l(t)}{du_{ij}}$$

where $\sigma_i(t) \equiv \sigma((2v_i(t+1) - 1)\sum_j w_{ij}\phi_j(t))$, $\phi_l'(t) \equiv d\phi_l(t)/dt$ and the hidden unit derivatives are found from the recursions

$$\frac{dh_l(t+1)}{du_{ij}} = \sum_k u_{lk}\frac{d\psi_k(t)}{du_{ij}} + \delta_{il}\psi_j(t), \quad \frac{d\psi_k(t)}{du_{ij}} = \sum_m \frac{\partial\psi_k(t)}{\partial h_m(t)}\frac{dh_m(t)}{du_{ij}}$$

We considered a network with the simple linear type influences, $\boldsymbol{\Psi}(t) \equiv \boldsymbol{\Phi}(t) \equiv \begin{pmatrix} \mathbf{h}(t) \\ \mathbf{v}(t) \end{pmatrix}$, and restricted connectivity $\mathbf{W} = \begin{pmatrix} \mathbf{A} & \mathbf{0} \\ \mathbf{0} & \mathbf{B} \end{pmatrix}$, $\mathbf{U} = \begin{pmatrix} \mathbf{C} & \mathbf{0} \\ \mathbf{0} & \mathbf{D} \end{pmatrix}$, where the

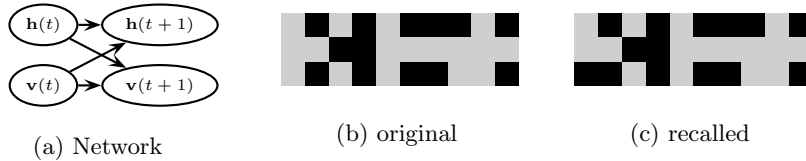

(a) Network        (b) original        (c) recalled

Figure 3: (a) A temporal slice of the network. (b) The training sequence consists of a random set vectors ($V = 3$) over $T = 10$ time steps. (c) The reconstruction using $H = 7$ hidden units. The initial state $\mathbf{v}(t = 1)$ for the recalled sequence was set to the correct initial training value albeit with one of the values flipped. Note how the dynamics learned is an attractor for the original sequence.

parameters to learn are the matrices $\mathbf{A}, \mathbf{B}, \mathbf{C}, \mathbf{D}$. A slice of the network is illustrated in fig(3)[a]. We can easily iterate the hidden states in this case to give

$$\mathbf{h}(t+1) = \mathbf{A}\mathbf{h}(t) + \mathbf{B}\mathbf{v}(t) = \mathbf{A}^t \mathbf{h}(1) + \sum_{t'=0}^{t-1} \mathbf{A}^{t'} \mathbf{B}\mathbf{v}(t - t')$$

which demonstrates how the hidden state depends on the full past history of the observations. We trained the network using 3 visible units and 7 hidden units to maximise the likelihood of the binary sequence in fig(3)[b]. Note that this sequence contains repeated patterns and therefore could not be recalled perfectly with a model which does not contain hidden units. We tested if the learned model had captured the dynamics of the training sequence by initialising the network in the first visible state in the training sequence, but with one of the values flipped. The network then generated the following hidden and visible states recursively, as plotted in fig(3)[c]. The learned network is an attractor with the training sequence as a stable point, demonstrating that such models are capable of learning attractor recurrent networks more powerful than those without hidden units. Learning is very fast in such networks, and we have successfully applied these models to cases of several hundred hidden and visible unit dimensions.

## 3.1    Recall Capacity

What effect have the hidden units on the ability of Hopfield networks to recall sequences? By recall, we mean that a training sequence is correctly generated by the network given that only the initial state of the training sequence is presented to the trained network. For the analysis here, we will consider the retrieval dynamics to be completely deterministic, thus if we concatenate both hidden $\mathbf{h}(t)$ and visible variables $\mathbf{v}(t)$ into the vector $\mathbf{x}(t)$ and consider the deterministic hidden function $f(y) \equiv \text{thresh}(y)$ which is 1 if $y > 0$ and zero otherwise, then

$$x_i(t+1) = \text{thresh} \sum_j M_{ij} x_j(t). \tag{8}$$

Here $M_{ij}$ are the elements of the weight matrix representing the transitions from time $t$ to time $t + 1$. A desired sequence $\tilde{\mathbf{x}}(1), \ldots, \tilde{\mathbf{x}}(T)$ can be recalled correctly if we can find a matrix $\mathbf{M}$ and real numbers $\epsilon_i(t)$ such that

$$\mathbf{M} \left[ \tilde{\mathbf{x}}(1), \ldots, \mathbf{x}(T-1) \right] = \left[ \boldsymbol{\epsilon}(2), \ldots, \boldsymbol{\epsilon}(T) \right]$$

where the $\epsilon_i(t)$ are arbitrary real numbers for which thresh$(\epsilon_i(t)) = \tilde{x}_i(t)$. This system of linear equations can be solved if the matrix $[\tilde{\mathbf{x}}(1), \ldots, \tilde{\mathbf{x}}(T-1)]$ has rank $T-1$. The use of hidden units therefore increases the length of temporal sequences that we can store by forming, during learning, appropriate hidden representations $\mathbf{h}(t)$ such that the vectors $\begin{pmatrix} \mathbf{h}(2) \\ \mathbf{v}(2) \end{pmatrix}, \ldots, \begin{pmatrix} \mathbf{h}(T) \\ \mathbf{v}(T) \end{pmatrix}$ form a linearly independent set. Such vectors are clearly possible to generate if the matrix $\mathbf{U}$ is full rank. Thus recall can be achieved if $(V + H) \geq T - 1$.

The reader might consider forming from a set of linearly dependent patterns $\mathbf{v}(1), \ldots, \mathbf{v}(T)$ a linearly independent is by injecting the patterns into a higher dimensional space, $\mathbf{v}(t) \rightarrow \hat{\mathbf{v}}(t)$ using a non-linear mapping. This would appear to dispense with the need to use hidden units. However, if the same pattern in the training set is repeated at different times in the sequence (as in fig(3)[b]), no matter how complex this non-linear mapping, the resulting vectors $\hat{\mathbf{v}}(1), \ldots, \hat{\mathbf{v}}(T)$ will be linearly dependent. This demonstrates that hidden units not only solve the linear dependence problem for non-repeated patterns, they also solve it for repeated patterns. They are therefore capable of sequence disambiguation since the hidden unit representations formed are dependent on the full history of the visible units.

# 4 A Continuous Visible Illustration

To illustrate the use of the framework to continuous visible variables, we consider the simple Gaussian visible CPT model

$$p(\mathbf{v}(t+1)|\mathbf{v}(t), \mathbf{h}(t)) = \exp\left(-\frac{1}{2\sigma^2}\left[\mathbf{v}(t+1) - \mathbf{g}\left(\mathbf{A}\mathbf{h}(t) - \mathbf{B}\mathbf{v}(t)\right)\right]^2\right)/(2\pi\sigma^2)^{V/2}$$

$$\mathbf{h}(t+1) = \mathbf{f}\left(\mathbf{C}\mathbf{h}(t) + \mathbf{D}\mathbf{v}(t)\right) \tag{9}$$

where the functions $f$ and $g$ are in general non-linear functions of their arguments. In the case that $f(x) \equiv x$, and $g(x) \equiv x$ this model is a special case of the Kalman filter[5]. Training of these models by learning $\mathbf{A}, \mathbf{B}, \mathbf{C}, \mathbf{D}$ ($\sigma^2$ was set to 0.02 throughout) is straightforward using the forward error propagation techniques outlined earlier in section (2).

## 4.1 Classifying Japanese vowels

This UCI machine learning test problem consists of a set of multi-dimensional times series. Nine speakers uttered two Japanese vowels /ae/ successively to form discrete time series with 12 LPC cepstral coefficients. Each utterance forms a time series $\mathcal{V}$ whose length is in the range $T = 7$ to $T = 29$ and each vector $\mathbf{v}(t)$ of the time series contains 12 cepstral coefficients. The training data consists of 30 training utterances for each of the 9 speakers. The test data contains 370 time series, each uttered by one of the nine speakers. The task is to assign each of the test utterances to the correct speaker.

We used the special settings $f(x) \equiv x$ and $g(x) \equiv x$ to see if such a simple network would be able to perform well. We split the training data into a 2/3 train and a 1/3 validation part, training then a set of 10 models for each of the 9 speakers, with hidden unit dimensions taking the values $H = 1, 2, \ldots, 10$ and using 20 training iterations of conjugate gradient learning[1]. For simplicity, we used the same number of hidden units for each of the nine speaker models. To classify a test utterance, we chose the speaker model which had the highest likelihood of generating the test utterance, using an error of 0 if the utterance was assigned to the correct speaker and an error of 1 otherwise. The errors on the validation set for these 10 models

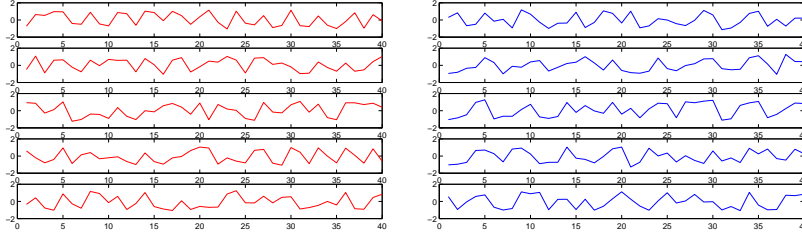

Figure 4: (Left)Five sequences from the model $v(t) = \sin(2(t-1) + \epsilon_1(t)) + 0.1\epsilon_2(t)$. (Right) Five sequences from the model $v(t) = \sin(5(t-1) + \epsilon_3(t)) + 0.1\epsilon_4(t)$, where $\epsilon_i(t)$ are zero mean unit variance Gaussian noise samples. These were combined to form a training set of 10 unlabelled sequences. We performed unsupervised learning by fitting a two component mixture model. The posterior probability $p(i = 1|\mathcal{V}^\mu)$ of the 5 sequences on the left belonging to class 1 are (from above) $0.99, 0.99, 0.83, 0.99, 0.96$ and for the 5 sequences on the right belonging to class 2 are (from above) $0.95, 0.99, 0.97, 0.97, 0.95$, in accord with the data generating process.

were $6, 6, 3, 5, 5, 5, 4, 5, 6, 3$. Based on these validation results, we retrained a model with $H = 3$ hidden units on all available training data. On the final independent test set, the model achieved an accuracy of 97.3%. This compares favourably with the 96.2% reported for training using a continuous-output HMM with 5 (discrete) hidden states[8]. Although our model is not powerful in being able to reconstruct the training data, it does learn sufficient information in the data to be able to make reliable classification. This problem serves to illustrate that such simple models can perform well. An interesting alternative training method not explored here would be to use discriminative learning[7]. Also, not explored here, is the possibility of using Bayesian methods to set the number of hidden dimensions.

## 5  Mixture Models

Since our models are probabilistic, we can apply standard statistical generalisations to them, including using them as part of a $M$ component mixture model

$$p(\mathcal{V}|\Theta) = \sum_{i=1}^{M} p(\mathcal{V}|\Theta_i, i) \, p(i) \tag{10}$$

where $p(i)$ denotes the prior mixing coefficients for model $i$, and each time series component model is represented by $p(\mathcal{V}|\Theta_i, i)$. Training mixture models by maximum likelihood on a set of sequences $\mathcal{V}^1, \ldots, \mathcal{V}^P$ is straightforward using the standard EM recursions [1]:

$$p^{new}(i) = \frac{\sum_{\mu=1}^{P} p(\mathcal{V}^\mu|i, \Theta_i^{old})p^{old}(i)}{\sum_{i=1}^{M} \sum_{\mu=1}^{P} p(\mathcal{V}^\mu|i, \Theta_i^{old})p^{old}(i)} \tag{11}$$

$$\Theta_i^{new} = \arg\max_{\Theta_i} \sum_{\mu=1}^{P} p(\mathcal{V}^\mu|i, \Theta_i^{old}) \log p(\mathcal{V}^\mu|i, \Theta_i) \tag{12}$$

To illustrate this on a simple example, we trained a mixture model with component models of the form described in section (4). The data is a series of 10 one dimensional ($V = 1$) time series each of length $T = 40$. Two distinct models were used

to generate 10 training sequences, see fig(4). We fitted a two component mixture model using mixture components of the form (9) (with linear functions $f$ and $g$) each model having $H = 3$ hidden units. After training, the model priors were found to be roughly equal $0.49, 0.51$ and it was satisfying to find that the separation of the unlabelled training sequences is entirely consistent with the data generation process, see fig(4). An interesting observation is that, whilst the true data generating process is governed by effectively stochastic hidden transitions, the deterministic hidden model still performs admirably.

## 6   Discussion

We have considered a class of models for temporal sequence processing which are a specially constrained version of Dynamic Bayesian Networks. The constraint was chosen to ensure that inference would be trivial even in high dimensional continuous hidden/latent spaces. Highly complex dynamics may therefore be postulated for the hidden space transitions, and also for the hidden to the visible transitions. However, unlike traditional neural networks the models remain probabilistic (generative models), and hence the full machinery of Bayesian inference is applicable to this class of models. Indeed, whilst not explored here, model selection issues, such as assessing the relevant hidden unit dimension, are greatly facilitated in this class of models. The potential use of this class of such models is therefore widespread. An area we are currently investigating is using these models for fast inference and learning in Independent Component Analysis and related areas. In the case that the hidden unit dynamics is known to be highly stochastic, this class of models is arguably less appropriate. However, stochastic hidden dynamics is often used in cases where one believes that the true hidden dynamics is too complex to model effectively (or, rather, deal with computationally) and one uses noise to 'cover' for the lack of complexity in the assumed hidden dynamics. The models outlined here provide an alternative in the case that a potentially complex hidden dynamics form can be assumed, and may also still provide a reasonable solution even in cases where the underlying hidden dynamics is stochastic. This class of models is therefore a potential route to computationally tractable, yet powerful time series models.

## References

[1] C.M. Bishop, *Neural Networks for Pattern Recognition*, Oxford University Press, 1995.

[2] H.A. Bourlard and N. Morgan, *Connectionist Speech Recognition. A Hybrid Approach.*, Kluwer, 1994.

[3] A. Doucet, N. de Freitas, and N. J. Gordon, *Sequential Monte Carlo Methods in Practice*, Springer, 2001.

[4] J. Hertz, A. Krogh, and R. Palmer, *Introduction to the theory of neural computation.*, Addison-Wesley, 1991.

[5] M. I. Jordan, *Learning in Graphical Models*, MIT Press, 1998.

[6] J.F. Kolen and S.C. Kramer, *Dynamic Recurrent Networks*, IEEE Press, 2001.

[7] A. Krogh and S.K. Riis, *Hidden Neural Networks*, Neural Computation **11** (1999), 541–563.

[8] M. Kudo, J. Toyama, and M. Shimbo, *Multidimensional Curve Classification Using Passing-Through Regions*, Pattern Recognition Letters **20** (1999), no. 11-13, 1103–1111.

[9] L.R. Rabiner and B.H. Juang, *An introduction to hidden Markov models*, IEEE Transactions on Acoustics Speech, Signal Processing **3** (1986), no. 1, 4–16.

[10] M. West and J. Harrison, *Bayesian forecasting and dynamic models*, Springer, 1999.
